# Recurrent Networks: Second Order Properties and Pruning

**Morten With Pedersen** and **Lars Kai Hansen**
CONNECT, Electronics Institute
Technical University of Denmark B349
DK-2800 Lyngby, DENMARK
emails: with,lkhansen@ei.dtu.dk

## Abstract

Second order properties of cost functions for recurrent networks are investigated. We analyze a layered fully recurrent architecture, the virtue of this architecture is that it features the conventional feedforward architecture as a special case. A detailed description of recursive computation of the *full Hessian* of the network cost function is provided. We discuss the possibility of invoking simplifying approximations of the Hessian and show how weight decays *iron* the cost function and thereby greatly assist training. We present tentative pruning results, using Hassibi et al.'s *Optimal Brain Surgeon*, demonstrating that recurrent networks can construct an efficient internal memory.

## 1  LEARNING IN RECURRENT NETWORKS

Time series processing is an important application area for neural networks and numerous architectures have been suggested, see e.g. (Weigend and Gershenfeld, 94). The most general structure is a fully recurrent network and it may be adapted using *Real Time Recurrent Learning* (RTRL) suggested by (Williams and Zipser, 89). By invoking a recurrent network, the *length* of the network memory can be adapted to the given time series, while it is fixed for the conventional lag-space net (Weigend et al., 90). In forecasting, however, feedforward architectures remain the most popular structures; only few applications are reported based on the Williams&Zipser approach. The main difficulties experienced using RTRL are slow convergence and

lack of generalization. Analogous problems in feedforward nets are solved using second order methods for training and pruning (LeCun et al., 90; Hassibi et al., 92; Svarer et al., 93). Also, regularization by weight decay significantly improves training and generalization. In this work we initiate the investigation of second order properties for RTRL; a detailed calculation scheme for the cost function Hessian is presented, the importance of weight decay is demonstrated, and preliminary pruning results using Hassibi et al.'s Optimal Brain Surgeon (OBS) are presented. We find that the recurrent network discards the available lag space and constructs its own efficient internal memory.

## 1.1  REAL TIME RECURRENT LEARNING

The fully connected feedback nets studied by Williams&Zipser operate like a state machine, computing the outputs from the internal units according to a state vector $\mathbf{z}(t)$ containing *previous* external inputs and internal unit outputs. Let $\mathbf{x}(t)$ denote a vector containing the external inputs to the net at time $t$, and let $\mathbf{y}(t)$ denote a vector containing the outputs of the units in the net. We now arrange the indices on $\mathbf{x}$ and $\mathbf{y}$ so that the elements of $\mathbf{z}(t)$ can be defined as

$$z_k(t) = \left\{ \begin{array}{ll} x_k(t) & , \quad k \in I \\ y_k(t) & , \quad k \in U \end{array} \right.$$

where $I$ denotes the set of indices for which $z_k$ is an input, and $U$ denotes the set of indices for which $z_k$ is the output of a unit in the net. Thresholds are implemented using an input permanently clamped to unity. The $k$'th unit in the net is now updated according to

$$y_k(t+1) = f_k[s_k(t)] = f_k \left[ \sum_{j \in I} w_{kj} x_j(t) + \sum_{j \in U} w_{kj} y_j(t) \right] = f_k \left[ \sum_{j \in I \cup U} w_{kj} z_j(t) \right]$$

where $w_{kj}$ denotes the weight to unit $k$ from input/unit $j$ and $f_k(\cdot)$ is the activation function of the $k$'th unit.

When used for time series prediction, the input vector (excluding threshold) is usually defined as $\mathbf{x}(t) = [x(t), \ldots, x(t-L+1)]$ where $L$ denotes the dimension of the *lag space*. One of the units in the net is designated to be the output unit $y_o$, and its activating function $f_o$ is often chosen to be linear in order to allow for arbitrary dynamical range. The prediction of $x(t+1)$ is $\hat{x}(t+1) = f_o[s_o(t)]$. Also, if the first prediction is at $t = 1$, the first example is presented at $t = 0$ and we set $\mathbf{y}(0) = \mathbf{0}$. We analyse here a modification of the standard Williams&Zipser construction that is appropriate for forecasting purposes. The studied architecture is *layered*. Firstly, we remove the external inputs from the linear output unit in order to prevent the network from getting trapped in a linear mode. The output then reads

$$\hat{x}(t+1) = y_o(t+1) = \sum_{j \in U} w_{oj} y_j(t) + w_{\text{thres},o} \tag{1}$$

Since $\mathbf{y}(0) = \mathbf{0}$ we obtain a first prediction yielding $\hat{x}(1) = w_{\text{thres},o}$ which is likely to be a poor prediction, and thereby introducing a significant error that is fed back into the network and used in future predictions. Secondly, when pruning

a fully recurrent feedback net we would like the net to be able to reduce to a simple two-layer feedforward net if necessary. Note that this is *not* possible with the conventional Williams&Zipser update rule, since it doesn't include a layered feedforward net as a special case. In a layered feedforward net the output unit is disconnected from the external inputs; in this case, cf. (1) we see that $\hat{x}(t+1)$ is based on the internal 'hidden' unit outputs $y_k(t)$ which are calculated on the basis of $\mathbf{z}(t-1)$ and thereby $\mathbf{x}(t-1)$. Hence, besides the startup problems, we also get a two-step ahead predictor using the standard architecture.

In order to avoid the problems with the conventional Williams&Zipser update scheme we use a layered updating scheme inspired by traditional feedforward nets, in which we distinguish between hidden layer units and the output unit. At time $t$, the hidden units work from the input vector $\mathbf{z}^h(t)$

$$
z_k^h(t) = \begin{cases} x_k(t-1) & , & k \in I \\ y_k^h(t-1) & , & k \in U \\ y^o(t-1) & , & k = O \end{cases}
$$

where $I$ denotes the input indices, $U$ denotes the hidden layer units and $O$ the output unit. Further, we use superscripts $h$ and $o$ to distinguish between hidden unit and output units. The activation of the hidden units is calculated according to

$$
y_k^h(t) = f_k^h[s_k^h(t)] = f_k^h \left[ \sum_{j \in I \cup U \cup O} w_{kj} z_j^h(t) \right] \quad , \quad k \in U \tag{2}
$$

The hidden unit outputs are forwarded to the output unit, which then sees the input vector $\mathbf{z}_k^o(t)$

$$
z_k^o(t) = \begin{cases} y_k^h(t) & , & k \in U \\ y^o(t-1) & , & k = O \end{cases}
$$

and is updated according to

$$
y^o(t) = f^o[s^o(t)] = f^o \left[ \sum_{j \in U \cup O} w_{oj} z_j^o(t) \right] \tag{3}
$$

The cost function is defined as $C = E + \mathbf{w}^T \mathbf{R} \mathbf{w}$. $\mathbf{R}$ is a regularization matrix, $\mathbf{w}$ is the concatenated set of parameters, and the sum of squared errors is

$$
E = \frac{1}{2} \sum_{t=1}^{T} [e(t)]^2 \quad , \quad e(t) = x(t) - y^o(t), \tag{4}
$$

where $T$ is the size of the training set series. RTRL is based on gradient descent in the cost function, here we investigate accelerated training using Newton methods. For that we need to compute first and second derivatives of the cost function. The essential difficulty is to determine derivatives of the sum of squared errors:

$$
\frac{\partial E}{\partial w_{ij}} = - \sum_{t=1}^{T} e(t) \frac{\partial y^o(t)}{\partial w_{ij}} \tag{5}
$$

The derivative of the output unit is computed as

$$\frac{\partial y^o(t)}{\partial w_{ij}} = \frac{\partial f^o[s^o(t)]}{\partial s^o(t)} \cdot \frac{\partial s^o(t)}{\partial w_{ij}} \tag{6}$$

where

$$\frac{\partial s^o(t)}{\partial w_{ij}} = \delta_{oi} z_j^o(t) + \sum_{j' \in U} w_{oj'} \frac{\partial y_{j'}^h(t)}{\partial w_{ij}} + w_{oo} \frac{\partial y^o(t-1)}{\partial w_{ij}} \tag{7}$$

where $\delta_{jk}$ is the Kronecker delta. This expression contains the derivative of the hidden units

$$\frac{\partial y_k^h(t)}{\partial w_{ij}} = \frac{\partial f_k^h[s_k^h(t)]}{\partial s_k^h(t)} \cdot \frac{\partial s_k^h(t)}{\partial w_{ij}} \quad , \quad k \in U \tag{8}$$

where

$$\frac{\partial s_k^h(t)}{\partial w_{ij}} = \delta_{ki} z_j^h(t) + \sum_{j' \in U} w_{kj'} \frac{\partial y_{j'}^h(t-1)}{\partial w_{ij}} + w_{ko} \frac{\partial y^o(t-1)}{\partial w_{ij}} \tag{9}$$

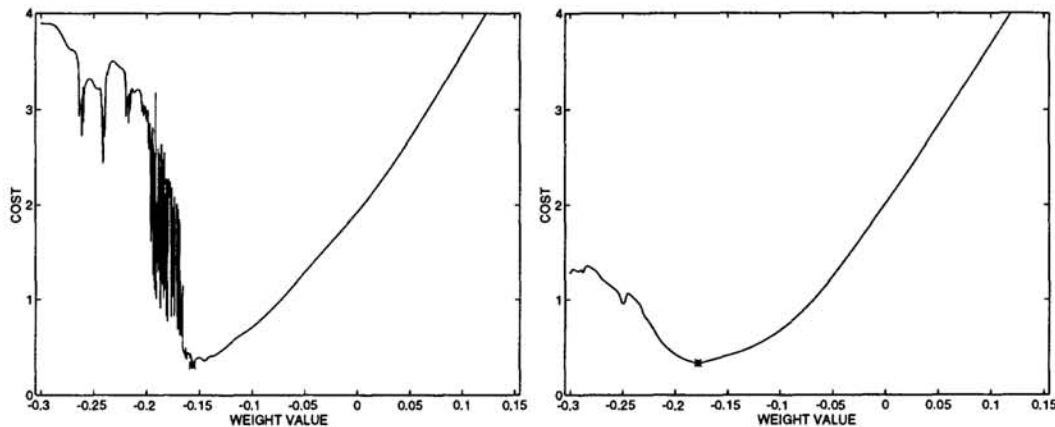

Figure 1: Cost function dependence of a weight connecting two hidden units for the sunspot benchmark series. Left panel: Cost function with small weight decay, the (local) optimum chosen is marked by an asterix. Right panel: The same slice through the cost function but here retrained with higher weight decay.

The complexity of the training problem for the recurrent net using RTRL is demonstrated in figure 1. The important role of weight decay (we have used a simple weight decay $\mathbf{R} = \alpha \mathbf{1}$) in controlling the complexity of the cost function is evident in the right panel of figure 1. The example studied is the sunspot benchmark problem (see e.g. (Weigend et al., 90) for a definition). First, we trained a network with the small weight decay and recorded the left panel result. Secondly, the network was retrained with increased weight decay and the particular weight connecting two hidden units was varied to produce the right panel result. In both cases all other weights remained fixed at their optimal values for the given weight decay. In addition to the complexity visible in these one-parameter slices of the cost function, the cost function is highly anisotropic in weight space and consequently the network Hessian is ill-conditioned. Hence, gradient descent is hampered by slow convergence.

## 2 SECOND ORDER PROPERTIES OF THE COST FUNCTION

To improve training by use of Newton methods and for use in OBS-pruning we compute the second derivative of the error functional:

$$\frac{\partial^2 E}{\partial w_{ij} \partial w_{pq}} = -\sum_{t=1}^{T} \left[ e(t) \frac{\partial^2 y^o(t)}{\partial w_{ij} \partial w_{pq}} - \frac{\partial y^o(t)}{\partial w_{ij}} \cdot \frac{\partial y^o(t)}{\partial w_{pq}} \right] \tag{10}$$

The second derivative of the output is

$$\frac{\partial^2 y^o(t)}{\partial w_{ij} \partial w_{pq}} = \frac{\partial^2 f^o[s^o(t)]}{\partial s^o(t)^2} \cdot \frac{\partial s^o(t)}{\partial w_{ij}} \cdot \frac{\partial s^o(t)}{\partial w_{pq}} + \frac{\partial f^o[s^o(t)]}{\partial s^o(t)} \cdot \frac{\partial^2 s^o(t)}{\partial w_{ij} \partial w_{pq}} \tag{11}$$

with

$$\frac{\partial^2 s^o(t)}{\partial w_{ij} \partial w_{pq}} = \delta_{oi} \frac{\partial z_j^o(t)}{\partial w_{pq}} + \sum_{j' \in U} w_{oj'} \frac{\partial^2 y_{j'}^h(t)}{\partial w_{ij} \partial w_{pq}} + w_{oo} \frac{\partial^2 y^o(t-1)}{\partial w_{ij} \partial w_{pq}} + \delta_{op} \frac{\partial z_q^o(t)}{\partial w_{ij}} \tag{12}$$

This expression contains the second derivative of the hidden unit outputs

$$\frac{\partial^2 y_k^h(t)}{\partial w_{ij} \partial w_{pq}} = \frac{\partial^2 f_k^h[s_k^h(t)]}{\partial s_k^h(t)^2} \cdot \frac{\partial s_k^h(t)}{\partial w_{ij}} \cdot \frac{\partial s_k^h(t)}{\partial w_{pq}} + \frac{\partial f_k^h[s_k^h(t)]}{\partial s_k^h(t)} \cdot \frac{\partial^2 s_k^h(t)}{\partial w_{ij} \partial w_{pq}} \tag{13}$$

with

$$\frac{\partial^2 s_k^h(t)}{\partial w_{ij} \partial w_{pq}} = \delta_{ki} \frac{\partial z_j^h(t)}{\partial w_{pq}} + \sum_{j' \in U} w_{kj'} \frac{\partial^2 y_{j'}^h(t-1)}{\partial w_{ij} \partial w_{pq}} + w_{ko} \frac{\partial^2 y^o(t-1)}{\partial w_{ij} \partial w_{pq}} + \delta_{kp} \frac{\partial z_q^h(t)}{\partial w_{ij}} \tag{14}$$

Recursion in the five index quantity (14) imposes a significant computational burden; in fact the first term of the Hessian in (10), involving the second derivative, is often neglected for computational convenience (LeCun et al., 90). Here we start by analyzing the significance of this term during training. We train a layered architecture to predict the sunspot benchmark problem. In figure 2 the ratio between the largest eigenvalue of the second derivative term in (10) and the largest eigenvalue of the full Hessian is shown. The ratio is presented for two different magnitudes of weight decay. In line with our observations above the second order properties of the "ironed" cost function are manageable, and we can simplify the Hessian calculation by neglecting the second derivative term in (10), i.e., apply the Gauss-Newton approximation.

## 3 PRUNING BY THE OPTIMAL BRAIN SURGEON

Pruning of recurrent networks has been pursued by (Giles and Omlin, 94) using a heuristic pruning technique, and significant improvement in generalization for a sequence recognition problem was demonstrated. Two pruning schemes are based on systematic estimation of weight *saliency*: the Optimal Brain Damage (OBD) scheme of (LeCun et al., 90) and OBS by (Hassibi et al., 93). OBD is based on the diagonal approximation of the Hessian and is very robust for forecasting (Svarer et al., 93). If an estimate of the full Hessian is available OBS can be used

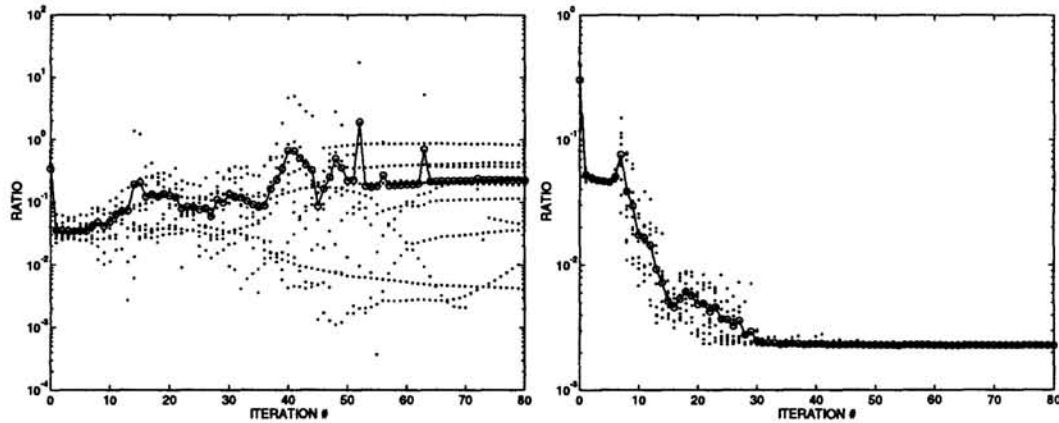

Figure 2: Ratio between the largest magnitude eigenvalue of the second derivative term of the Hessian (c.f. equation (10)) and the largest magnitude eigenvalue of the complete Hessian as they appeared during ten training sessions. The connected circles represent the average ratio. Left panel: Training with small weight decay. Right panel: Training with a high weight decay.

for estimation of saliencies incorporating *linear* retraining. In (Hansen and With Pedersen, 94) OBS was generalized to incorporate weight decays; we use these modifications in our experiments. Note that OBS in its standard form only allows for one weight to be eliminated at a time. The result of a pruning session is a nested family of networks. In order to select the optimal network within the family it was suggested in (Svarer et al., 93.) to use the estimated test error. In particular we use Akaike's Final Prediction Error (Akaike, 69) to estimate the network test error $\widehat{E}_{\text{test}} = ((T+N)/(T-N)) \cdot 2E/T$ [1], and $N$ is the number of parameters in the network. In figure 3 we show the results of such a pruning session on the sunspot data starting from a (4-4-1) network architecture. The recurrent network was trained using a *damped* Gauss-Newton scheme. Note that the training error increases as weights are eliminated, while the test error and the estimated test error both pass through shallow minima showing that generalization is slightly improved by pruning. In fact, by retraining the optimal architecture with reduced weight decay both training and test errors are decreased in line with the observations in (Svarer et al., 93). It is interesting to observe that the network, though starting with access to a lag-space of four delay units, has lost three of the delayed inputs; hence, rely solely on its internal memory, as seen in the right panel of figure 3. To further illustrate the memory properties of the optimal network, we show in figure 4 the network response to a unit impulse. It is interesting that the response of the network extends for approximately 12 time steps corresponding to the "period" of the sunspot series.

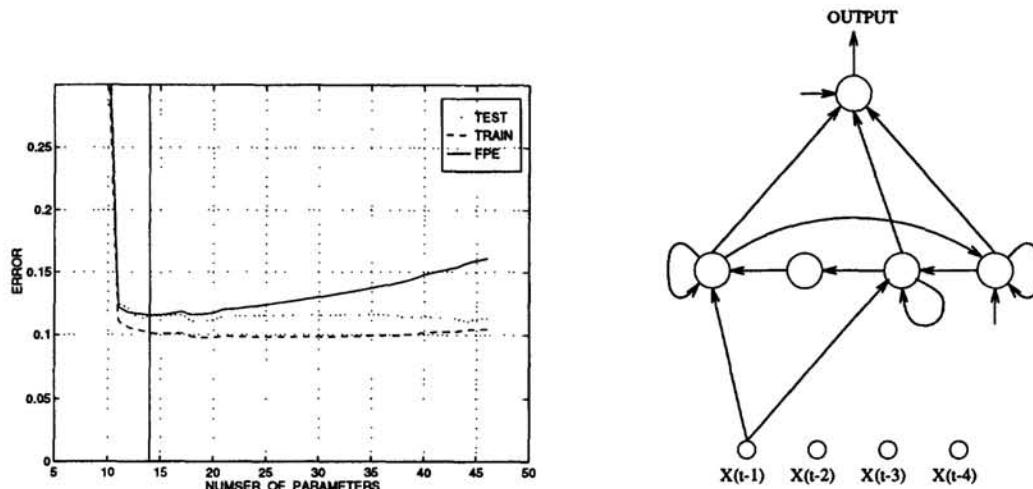

Figure 3: Left panel: OBS pruning of a (4-4-1) recurrent network trained on sunspot benchmark. Development of training error, test error, and Akaike estimated test error (FPE). Right panel: Architecture of the FPE-optimal network. Note that the network discards the available lag space and solely predicts from internal memory.

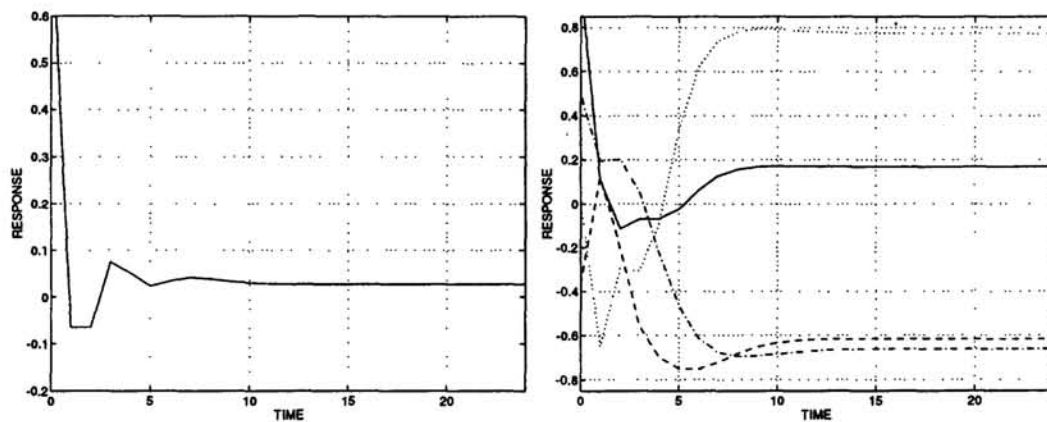

Figure 4: Left panel: Output of the pruned network after a unit impulse input at $t = 0$. The internal memory is about 12 time units long which is, in fact, roughly the period of the sunspot series. Right panel: Activity of the four hidden units in the pruned network after a unit impulse at time $t = 0$.

## 4 CONCLUSION

A layered recurrent architecture, which has a feedforward net as a special case, has been investigated. A scheme for recursive estimation of the Hessian of the fully recurrent neural net is devised. It's been shown that weight decay plays a decisive role when adapting recurrent networks. Further, it is shown that the second order information may be used to train and prune a recurrent network and in this process the network may discard the available lag space. The network builds an efficient

internal memory extending beyond the lag space that was originally available.

## Acknowledgments

We thank Jan Larsen, Sara Solla, and Claus Svarer for useful discussions, and Lee Giles for providing us with a preprint of (Giles and Omlin, 94). We thank the anonymous reviewers for valuable comments on the manuscript. This research is supported by the Danish Natural Science and Technical Research Councils through the Computational Neural Network Center (CONNECT).

## Footnotes

[1] The use of Akaike's estimate is not well justified for a feedback net, test error estimates for feedback models is a topic of current research.

## References

H. Akaike: *Fitting Autoregressive Models for Prediction.* Ann. Inst. Stat. Mat. **21**, 243-247, (1969).

Y. Le Cun, J.S. Denker, and S.A. Solla: *Optimal Brain Damage.* In Advances in Neural Information Processing Systems 2, (Ed. D.S. Touretzsky) Morgan Kaufmann, 598-605, (1990).

C.L. Giles and C.W. Omlin: *Pruning of Recurrent Neural Networks for Improved Generalization Performance.* IEEE Transactions on Neural Networks, to appear. Preprint NEC Research Institute (1994).

L.K. Hansen and M. With Pedersen: *Controlled Growth of Cascade Correlation Nets*, International Conference on Artificial Neural Networks ICANN'94 Sorrento. (Eds. M. Marinaro and P.G. Morasso) Springer, 797-801, (1994).

B. Hassibi, D. G. Stork, and G. J. Wolff, *Optimal Brain Surgeon and General Network Pruning*, in Proceedings of the 1993 IEEE International Conference on Neural Networks, San Francisco (Eds. E.H. Ruspini et al. ) IEEE, 293-299 (1993).

C. Svarer, L.K. Hansen, and J. Larsen: *On Design and Evaluation of Tapped Delay Line Networks*, In Proceedings of the 1993 IEEE International Conference on Neural Networks, San Francisco, (Eds. E.H. Ruspini et al. ) 46-51, (1993).

A.S. Weigend, B.A. Huberman, and D.E. Rumelhart: *Predicting the future: A Connectionist Approach.* Int. J. of Neural Systems **3**, 193-209 (1990).

A.S. Weigend and N.A. Gershenfeld, Eds.: *Times Series Prediction: Forecasting the Future and Understanding the Past.* Redwood City, CA: Addison-Wesley (1994).

R.J. Williams and D. Zipser: *A Learning Algorithm for Continually Running Fully Recurrent Neural Networks*, Neural Computation **1**, 270-280, (1989).